# Extending Phase Mechanism to Differential Motion Opponency for Motion Pop-Out

**Yicong Meng and Bertram E. Shi**
Department of Electronic and Computer Engineering
Hong Kong University of Science and Technology
Clear Water Bay, Kowloon, Hong Kong
{eeyicong, eebert}@ust.hk

## Abstract

We extend the concept of phase tuning, a ubiquitous mechanism among sensory neurons including motion and disparity selective neurons, to the motion contrast detection. We demonstrate that the motion contrast can be detected by phase shifts between motion neuronal responses in different spatial regions. By constructing the differential motion opponency in response to motions in two different spatial regions, varying motion contrasts can be detected, where similar motion is detected by zero phase shifts and differences in motion by non-zero phase shifts. The model can exhibit either enhancement or suppression of responses by either different or similar motion in the surrounding. A primary advantage of the model is that the responses are selective to relative motion instead of absolute motion, which could model neurons found in neurophysiological experiments responsible for motion pop-out detection.

## 1    Introduction

Motion discontinuity or motion contrast is an important cue for the pop-out of salient moving objects from contextual backgrounds. Although the neural mechanism underlying the motion pop-out detection is still unknown, the center-surround receptive field (RF) organization is considered as a physiological basis responsible for the pop-out detection.

The center-surround RF structure is simple and ubiquitous in cortical cells especially in neurons processing motion and color information. Nakayama and Loomis [1] have predicted the existence of motion selective neurons with antagonistic center-surround receptive field organization in 1974. Recent physiological experiments [2][3] show that neurons with center-surround RFs have been found in both middle temporal (MT) and medial superior temporal (MST) areas related to motion processing. This antagonistic mechanism has been suggested to detect motion segmentation [4], figure/ground segregation [5] and the differentiation of object motion from ego-motion [6].

There are many related works [7]-[12] on motion pop-out detection. Some works [7]-[9] are based on spatio-temporal filtering outputs, but motion neurons are not fully interacted by either only inhibiting similar motion [7] or only enhancing opposite motion [8]. Heeger, et al. [7] proposed a center-surround operator to eliminate the response dependence upon rotational motions. But the Heeger's model only shows a complete center-surround interaction for moving directions. With respect to the surrounding speed effects, the neuronal responses are suppressed by the same speed with the center motion but not enhanced by other speeds. Similar problem existed in [8], which only modeled the suppression of neuronal responses in the classical receptive field (CRF) by similar motions in surrounding regions. Physiological experiments [10][11] show that many neurons in visual cortex are sensitive to the motion contrast rather than depend upon the absolute direction and speed of the object motion. Although pooling over motion neurons tuned to different velocities can

eliminate the dependence upon absolute velocities, it is computationally inefficient and still can't give full interactions of both suppression and enhancement by similar and opposite surrounding motions. The model proposed by Dellen, et al. [12] computed differential motion responses directly from complex cells in V1 and didn't utilize responses from direction selective neurons.

In this paper, we propose an opponency model which directly responds to differential motions by utilizing the phase shift mechanism. Phase tuning is a ubiquitous mechanism in sensory information processing, including motion, disparity and depth detection. Disparity selective neurons in the visual cortex have been found to detect disparities by adjusting the phase shift between the receptive field organizations in the left and right eyes [13][14]. Motion sensitive cells have been modeled in the similar way as the disparity energy neurons and detect image motions by utilizing the phase shift between the real and imaginary parts of temporal complex valued responses, which are comparable to images to the left and right eyes [15]. Therefore, the differential motion can be modeled by exploring the similarity between images from different spatial regions and from different eyes.

The remainder of this paper is organized as following. Section 2 illustrates the phase shift motion energy neurons which estimate image velocities by the phase tuning in the imaginary path of the temporal receptive field responses. In section 3, we extend the concept of phase tuning to the construction of differential motion opponency. The phase difference determines the preferred velocity difference between adjacent areas in retinal images. Section 4 investigates properties of motion pop-out detection by the proposed motion opponency model. Finally, in section 5, we relate our proposed model to the neural mechanism of motion integration and motion segmentation in motion related areas and suggest a possible interpretation for adaptive center-surround interactions observed in biological experiments.

## 2      Phase Shift Motion Energy Neurons

Adelson and Bergen [16] proposed the motion energy model for visual motion perception by measuring spatio-temporal orientations of image sequences in space and time. The motion energy model posits that the responses of direction-selective V1 complex cells can be computed by a combination of two linear spatio-temporal filtering stages, followed by squaring and summation. The motion energy model was extended in [15] to be phase tuned by splitting the complex valued temporal responses into real and imaginary paths and adding a phase shift on the imaginary path.

Figure 1(a) demonstrates the schematic diagram of the phase shift motion energy model. Here we assume an input image sequence in two-dimensional space $(x, y)$ and time $t$. The separable spatio-temporal receptive field ensures the cascade implementation of RF with spatial and temporal filters. Due to the requirement of the causal temporal RF, the phase shift motion energy model didn't adopt the Gabor filter like the spatial RF. The phase shift spatio-temporal RF is modeled with a complex valued function $f(x,y,t) = g(x,y) \cdot h(t,\Phi)$, where the spatial and temporal RFs are denoted by $g(x,y)$ and $h(t,\Phi)$ respectively,

$$
\begin{aligned}
g(x,y) &= \mathcal{N}(x,y \mid 0,\mathbf{C})\exp\left(j\Omega_x x + j\Omega_y y\right) \\
h(t,\Phi) &= h_{\text{real}}(t) + \exp(j\Phi)h_{\text{imag}}(t)
\end{aligned}
\tag{1}
$$

and $\mathbf{C}$ is the covariance matrix of the spatial Gaussian envelope and $\Phi$ is the phase tuning of the motion energy neuron. The real and imaginary profiles of the temporal receptive field are Gamma modulated sinusoidal functions with quadrature phases,

$$
\begin{aligned}
h_{\text{real}}(t) &= \mathcal{G}(t \mid \alpha,\tau)\cos(\Omega_t t) \\
h_{\text{imag}}(t) &= \mathcal{G}(t \mid \alpha,\tau)\sin(\Omega_t t)
\end{aligned}
\tag{2}
$$

The envelopes for complex exponentials are functions of Gaussian and Gamma distributions,

$$
\mathcal{N}(x,y \mid 0,\mathbf{C}) = \frac{1}{2\pi\sigma_x\sigma_y}\exp\left(-\frac{x^2}{2\sigma_x^2} - \frac{y^2}{2\sigma_y^2}\right)
\tag{3}
$$

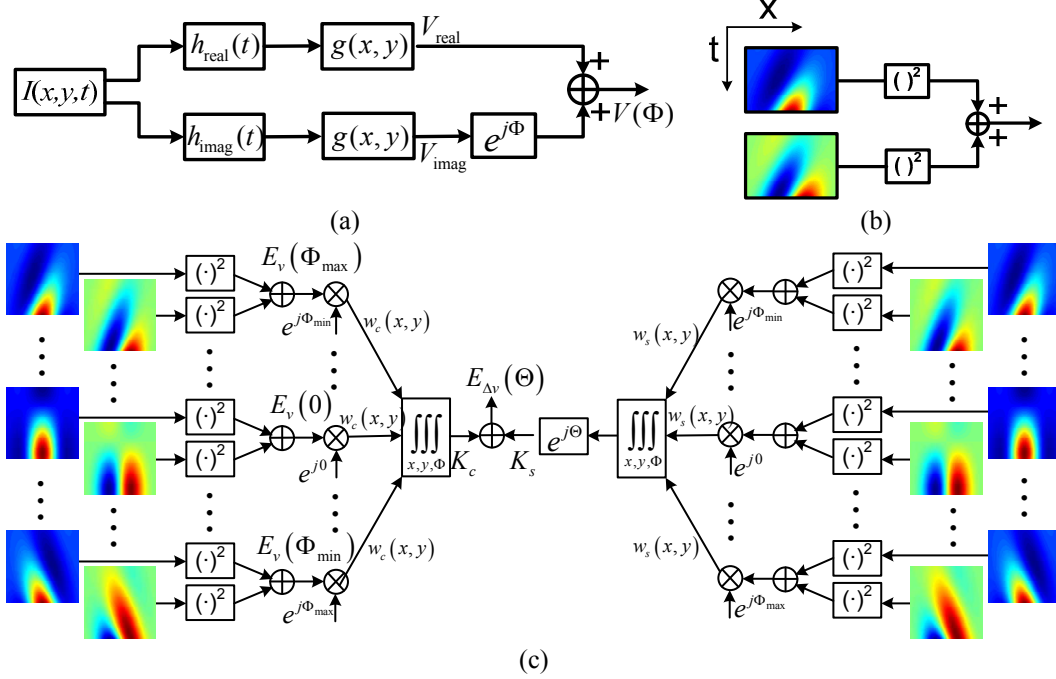

(a)

(b)

(c)

Figure 1. (a) shows the diagram of the phase shift motion energy model adapted from [15]. (b) draws the spatiotemporal representation of the phase shift motion energy neuron with the real and imaginary receptive field demonstrated by the two left pictures. (c) illustrates the construction of differential motion opponency with a phase difference $\Theta$ from two populations of phase shift motion energy neurons in two spatial areas $c$ and $s$. To avoid clutter, the space location $(x, y)$ is not explicitly shown in phase tuned motion energies.

$$G(t \mid \alpha, \tau) = \frac{1}{\Gamma(\alpha)\tau^{\alpha}} t^{\alpha-1} \exp\left(-\frac{t}{\tau}\right) u(t) \tag{4}$$

where $\Gamma(\alpha)$ is the gamma function and $u(t)$ is the unit step function. The parameters $\alpha$ and $\tau$ determine the temporal RF size. As derived in [15], the motion energy at location $(x, y)$ can be computed by

$$E_v(x, y, \Phi) = S + P\cos(\Psi - \Phi) \tag{5}$$

where

$$\begin{aligned}
S &= \left\|V_{\text{real}}\right\|^2 + \left\|V_{\text{imag}}\right\|^2 \\
P &= 2\left\|V_{\text{real}}V_{\text{imag}}^*\right\| \\
\Psi &= \arg\left(V_{\text{real}}V_{\text{imag}}^*\right)
\end{aligned} \tag{6}$$

and complex valued responses in real and imaginary paths are obtained as,

$$\begin{aligned}
V_{\text{real}}(x, y, t) &= \iiint_{\xi,\zeta,\eta} g(\xi, \zeta) h_{\text{real}}(\eta) I(x-\xi, y-\zeta, t-\eta) d\xi d\zeta d\eta \\
V_{\text{imag}}(x, y, t) &= \iiint_{\xi,\zeta,\eta} g(\xi, \zeta) h_{\text{imag}}(\eta) I(x-\xi, y-\zeta, t-\eta) d\xi d\zeta d\eta
\end{aligned} \tag{7}$$

The superscript * represents the complex conjugation and the phase shift parameter $\Phi$ controls the spatio-temporal orientation tuning. To avoid clutter, the spatial location variables $x$ and $y$ for $S$, $P$, $\Psi$, $V_{\text{real}}$ and $V_{\text{imag}}$ are not explicitly shown in Eq. (5) and (6). Figure 1(b) demonstrates the even and odd profiles of the spatio-temporal RF tuned to a particular phase shift.

$\Theta < 0$

$\Theta > 0$

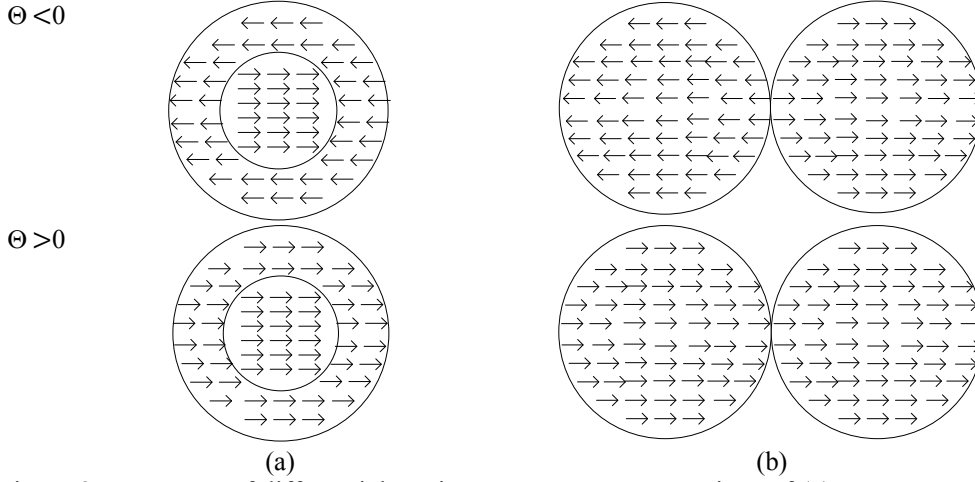

(a)                                        (b)

Figure 2. Two types of differential motion opponency constructions of (a) center-surrounding interaction and (b) left-right interaction. Among cells in area MT with surrounding modulations, 25% of cells are with the antagonistic RF structure in the top row and another 50% of cells have the integrative RF structure as shown in the bottom row.

## 3   Extending Phase Mechanism to Differential Motion Opponency

Based on the above phase shift motion energy model, the local image velocity at each spatial location can be represented by a phase shift which leads to the peak response across a population of motion energy neurons. Across regions of different motions, there are clear discontinuities on the estimated velocity map. The motion discontinuities can be detected by edge detectors on the velocity map to segment different motions. However, this algorithm for motion discontinuities detection can't discriminate between the object motion and uniform motions in contextual backgrounds.

Here we propose a phase mechanism to detect differential motions inspired by the disparity energy model and adopt the center-surround inhibition mechanism to pop out the object motion from contextual background motions. The motion differences between different spatial locations can be modeled in the similar way as the disparity model. The motion energies from two neighboring locations are considered as the retinal images to the left and right eyes. Thus, we can construct a differential motion opponency by placing two populations of phase shift motion energy neurons at different spatial locations and the energy $E_{\Delta v}(\Theta)$ of the opponency is the squared modulus of the averaged phase shift motion energies over space and phase,

$$\mathrm{E}_{\Delta v}(\Theta) = \left\| \iiint E_v(x, y, \Phi) \cdot w(x, y, \Phi \mid \Theta) \, dx dy d\Phi \right\|^2 \qquad (8)$$

where $w(x, y, \Theta)$ is the profile for differential motion opponency and $\Delta v$ is the velocity difference between the two spatial regions defined by the kernel $w(x, y, \Theta)$. Since $w(x, y, \Theta)$ is intended to implement the functional role of spatial interactions, it is desired to be a separable function in space and phase domain and can be modeled by phase tuned summation of two spatial kernels,

$$w(x, y, \Phi \mid \Theta) = w_c(x, y)e^{j\Phi} + e^{j\Theta + j\Phi} w_s(x, y) \qquad (9)$$

where $w_c(x, y)$ and $w_s(x, y)$ are Gaussian kernels of different spatial sizes $\sigma_c$ and $\sigma_s$, and $\Theta$ is the phase difference representing velocity difference between two spatial regions $c$ and $s$. Substituting Eq. (9) into Eq. (8), the differential motion energy can be reformulated as

$$E_{\Delta v}(\Theta) = \left\| K_c + e^{j\Theta} K_s \right\|^2 \qquad (10)$$

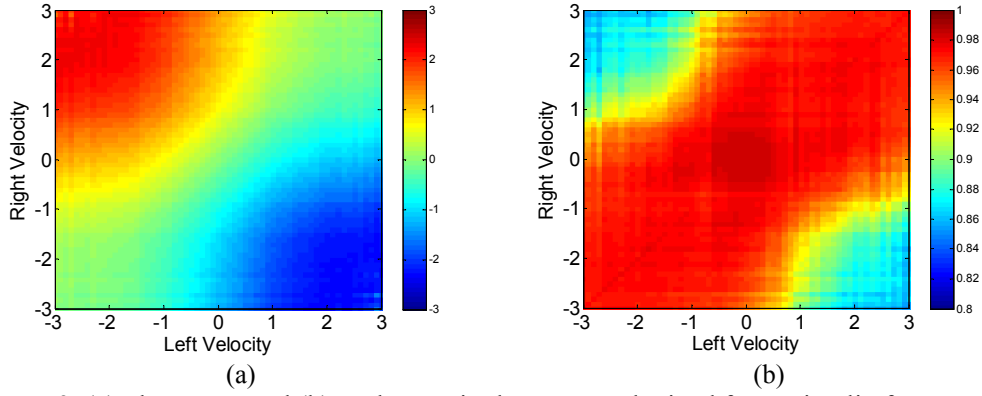

(a)                                                  (b)

Figure 3. (a) Phase map and (b) peak magnitude map are obtained from stimuli of two patches of random dots moving with different velocities. The two patches of stimuli are statistically independent but share the same spatial properties: dot size of 2 pixels, dot density of 10% and dot coherence level of 100%. The phase tuned population of motion energy neurons are applied to each patch of random dots with RF parameters: $\Omega_t = 2\pi/8$, $\Omega_t = 2\pi/16$, $\sigma_x = 5$ and $\tau = 5.5$. For each combination of velocities from left and right patches, averaged phase shifts over space and time are computed and so do the magnitudes of peak responses. The unit for velocities is pixels per frame.

where

$$K_c = \iiint_{x,y,\Phi} E_{v,c}(x,y,\Phi)\exp(j\Phi)w_c(x,y)dxdyd\Phi$$

$$K_s = \iiint_{x,y,\Phi} E_{v,s}(x,y,\Phi)\exp(j\Phi)w_s(x,y)dxdyd\Phi$$

(11)

$E_{v,c}(x,y,\Phi)$ and $E_{v,s}(x,y,\Phi)$ are phase shift motion energies at location $(x,y)$ and with phase shift $\Phi$. Utilizing the results in Eq. (5) and (6), Eq. (10) and (11) generate similar results,

$$\mathrm{E}_{\Delta v}(\Theta) = S_{\mathrm{opp}} + P_{\mathrm{opp}}\cos(\Theta_{\mathrm{opp}} - \Theta)$$

(12)

where

$$S_{\mathrm{opp}} = \|K_c\|^2 + \|K_s\|^2$$

$$P_{\mathrm{opp}} = 2\|K_c K_s^*\|$$

$$\Theta_{\mathrm{opp}} = \arg(K_c K_s^*)$$

(13)

According to above derivations, by varying the phase shift $\Theta$ between $-\pi$ and $\pi$, the relative motion energy of the differential motion opponency can be modeled as population responses across a population of phase tuned motion opponencies. The response is completely specified by three parameters $S_{\mathrm{opp}}$, $P_{\mathrm{opp}}$ and $\Theta_{\mathrm{opp}}$.

The schematic diagram of this opponency is illustrated in Figure 1(c). The differential motion opponency is constituted by three stages. At the first stage, a population of phase shift motion energy neurons is applied to be selective to different velocities. At the second stage, motion energies from the first stage are weighted by kernels tuned to different spatial locations and phase shifts respectively for both spatial regions and two single differential motion signals in region $c$ and region $s$ are achieved by integrating responses from these two regions over space and phase tuning. Finally, the differential motion energy is computed by the squared modulus of the summation of the integrated motion signal in region $c$ and phase shifted motion signal in region $s$. The subscripts $c$ and $s$ represent two interacted spatial regions which are not limited to the center and surround regions. The opponency could also be constructed by the neighboring left and right

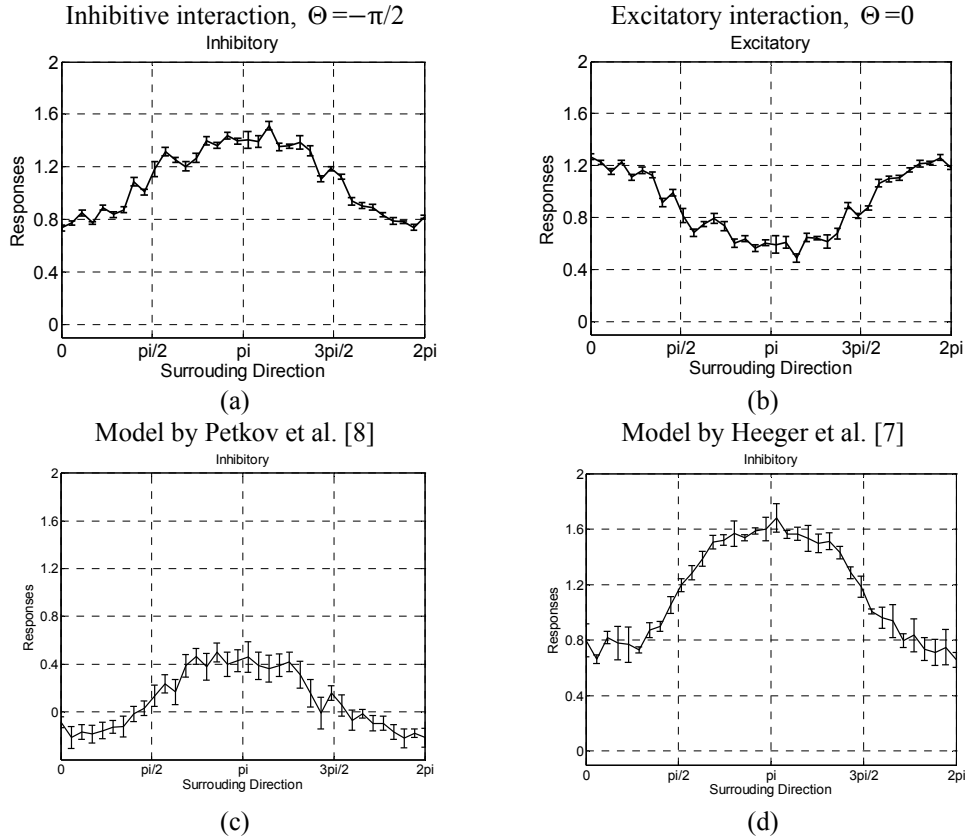

Figure 4. Demonstrations of center-surround differential motion opponency, where (a) show the excitation of opposite directions outside the CRF and (b) show the inhibition by surrounding motions in same directions. The center-surround inhibition models by Petkov, et al. [8] and Heeger, et al. [7] are shown in (c) and (d). Responses above 1 indicate enhancement and responses below 1 indicate suppressions.

spatial regions. Figure 2 shows two types of structures for the differential motion opponency. In [17], the authors demonstrates that among cells in area MT with surrounding modulations, 25% of cells are with the antagonistic RF structure as shown in Figure 2(a) and another 50% of cells have the integrative RF structure as shown in Figure 2(b).

The velocity difference tuning of the opponency is determined by the phase shift parameter $\Theta$ combined with parameters of spatial and temporal frequencies for motion energy neurons. The larger phase shift magnitude prefers the bigger velocity difference. This phase tuning of velocity difference is consistent with the phase tuning of motion energy neurons. Figure 3 shows the phase map obtained by using random dots stimuli with different velocities on two spatial patches (left and right patches with sizes of 128 pixels $\times$ 128 pixels). Along the diagonal line, velocities from left and right patches are equal to each other and therefore phase estimates are zeros along this line. Deviated from the diagonal line to upper-left and lower-right, the phase magnitudes increase while positive phases indicate larger left velocities and negative phases indicate larger right velocities. The phase tuning can give a good classification of velocity differences.

## 4    Validation of Differential Motion Opponency

Out derivation and analysis above show that the phase shift between two neighboring spatial regions is a good indicator for motion difference between these two regions. In this section, we validate the proposed differential motion opponency by two sets of experiments, which show effects of both surrounding directions and speeds on the center motion.

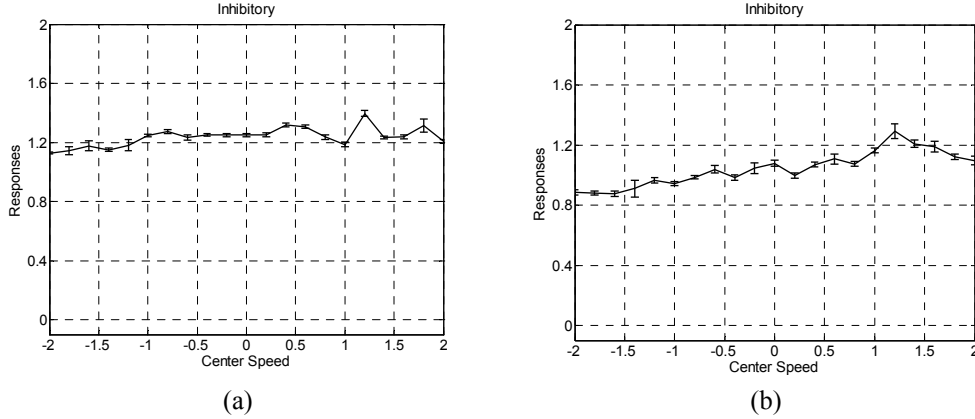

(a)                              (b)

Figure 5. The insensitivity of the proposed opponency model to absolute center and surrounding velocities is demonstrated in (a), where responses are enhanced for all center velocities from -2 to 2 pixels per frame. In (b), the model by Heeger, et al. [7] only shows enhancement when the center speed matches the preferred speed of 1.2 pixel per frame. Similarly, responses above 1 indicate enhancement and below 1 indicate suppressions. In both curves, the velocity differences between center and surrounding regions are maintained as a constant of 3 pixels per frame.

Physiological experiments [2][3] have demonstrated that the neuronal activities in the classical receptive field are suppressed by responses outside the CRF to stimuli with similar motions including both directions and speeds on the center and surrounding regions. On the contrary, visual stimuli of opposite directions or quite different speeds outside the CRF enhance the responses in the CRF. In their experiments, they used a set of stimuli of random dots moving at different velocities, where there are small patches of moving random dots on the center.

We tested the properties of the proposed opponency model for motion difference measurement by using similar random dots stimuli. The random dots on background move with different speeds and in different direction but have the same statistical parameters: dot size of 2 pixels, dot density of 10% and motion coherence level of 100%. The small random dots patches are placed on the center of background stimuli to stimulate the neurons in the CRF. These small patches share the same statistical parameters with background random dots but move with a constant velocity of 1 pixel per frame.

Figure 4 shows results for the enhanced and suppressed responses in the CRF with varying surrounding directions. The phase shift motion energy neurons had the same spatial and temporal frequencies and the same receptive field sizes, and were selective to vertical orientations. The preferred spatial frequency was $2\pi/16$ radian per pixel and the temporal frequency was $2\pi/16$ radian per frame. The sizes of RF in horizontal and vertical directions were respectively 5 pixels and 10 pixels, corresponding to a spatial bandwidth of 1.96 octaves. The time constant $\tau$ was 5.5 frames which resulted in a temporal bandwidth of 1.96 octaves. As shown in Figure 4 (a) and (b), the surrounding motion of opposite direction gives the largest response to the motion in the CRF for the inhibitory interaction and the smallest response for the excitatory interaction.

Results demonstrated in Figure 4 are consistent with physiological results reported in [3]. In Born's paper, inhibitory cells show response enhancement and excitatory cells show response suppression when surrounding motions are in opposite directions. The 3-dB bandwidth for the surrounding moving direction is about 135 degrees for the physiological experiments while the bandwidth is about 180 degrees for the simulation results in our proposed model.

Models proposed by Petkov, et al. [8] and Heeger, et al. [7] also show clear inhibition between opposite motions. The Petkov's model achieves the surrounding suppression for each point in $(x, y, t)$ space by the subtraction between responses from that point and its surroundings and followed by a half-wave rectification,

$$\tilde{E}_{v,\theta}(x,y,t) = \left| E_{v,\theta}(x,y,t) - \alpha \cdot S_{v,\theta}(x,y,t) \right|^{+} \qquad (14)$$

where $E_{v,\theta}(x,y,t)$ is the motion energy at location $(x,y)$ and time $t$ for a given preferred speed $v$ and orientation $\theta$, $S_{v,\theta}(x,y,t)$ is the average motion energy in the surrounding of point $(x, y, t)$, $\tilde{E}_{v,\theta}(x,y,t)$ is the suppressed motion energy and the factor $\alpha$ controls the inhibition strength. The inhibition term is computed by weighted motion energy

$$S_{v,\theta}(x,y,t) = E_{v,\theta}(x,y,t) * w_{v,\theta}(x,y,t) \tag{15}$$

where $w_{v,\theta}(x,y,t)$ is the surround weighting function.

The Heeger's model constructs the center-surround motion opponent by computing the weighted sum of responses from motion selective cells,

$$R_{v,\theta}(t) = \sum_{x,y} \beta(x,y) \left[ E_{v,\theta}(x,y,t) - E_{-v,\theta}(x,y,t) \right] \tag{16}$$

where $\beta(x,y)$ is a center-surround weighting function and the motion energy at each point should be normalized across all cells with different tuning properties.

As shown in Figure 4 (c) and (d) for results of Petkov's and Heeger's models, we replace the conventional frequency tuned motion energy neuron with our proposed phase tuned neuron. The model by Petkov, et al. [8] is generally suppressive and only reproduces less suppression for opposite motions, which is inconsistent with results from [3]. The model by Heeger, et al. [7] has similar properties with our proposed model with respect to both excitatory and inhibitory interactions.

To evaluate the sensitivity of the proposed opponency model to velocity differences, we did simulations by using similar stimuli with the above experiment in Figure 4 but maintaining a constant velocity difference of 3 pixels per frame between the center and surrounding random dot patches. As shown in Figure 5, by varying the velocities of random dots on the center region, we found that responses by the proposed model are always enhanced independent upon absolute velocities of center stimuli, but responses by the Heeger's model achieve the enhancement at a center velocity of 1.2 pixels per frame and maintain suppressed at other speeds.

## 5    Discussion

We proposed a new biologically plausible model of the differential motion opponency to model the spatial interaction property of motion energy neurons. The proposed opponency model is motivated by the phase tuning mechanism of disparity energy neurons which infers the disparity information from the phase difference between complex valued responses to left and right retinal images. Hence, the two neighboring spatial areas can be considered as left and right images and the motion difference between these two spatial regions is detected by the phase difference between the complex valued responses at these two regions. Our experimental results demonstrate a consistent conclusion with physiological experiments that motions of opposite directions and different speeds outside the CRF can show both inhibitive and excitatory effects on the CRF responses. The inhibitive interaction helps to segment the moving object from backgrounds when fed back to low-level features such as edges, orientations and color information.

Except providing a unifying phase mechanism in understanding neurons with different functional roles at different brain areas, the proposed opponency model could possibly provide a way to understand the motion integration and motion segmentation. Integration and segmentation are two opposite motion perception tasks but co-exist to constitute two fundamental types of motion processing. Segmentation is achieved by discriminating motion signals from different objects, which is thought to be due to the antagonistic interaction between center and surrounding RFs. Integration is obtained by utilizing the enhancing function of surrounding areas to CRF areas. Both types of processing have been found in motion related areas including area MT and MST. Tadin, et al. [18] have found that motion segmentation dominants at high stimulus contrast and gives the way to motion integration at low stimulus contrast. Huang, et al. [19] suggests that the surrounding modulation is adaptive according to the visual stimulus such as contrasts and noise levels. Since our proposed opponency model determines the functional role of neurons by only the phase shift parameter, this makes the proposed model to be an ideal candidate model for the adaptive surrounding modulation with the phase tuning between two spatial regions.

# References

[1]. K. Nakayama and J. M. Loomis, "Optical velocity patterns, velocity-sensitive neurons, and space perception: A hypothesis," *Perception*, vol. 3, 63-80, 1974.

[2]. K. Tanaka, K. Hikosaka, H. Saito, M. Yukie, Y. Fukada and E. Iwai, "Analysis of local and wide-field movements in the superior temporal visual areas of the macaque monkey," *Journal of Neuroscience*, vol. 6, pp. 134-144, 1986.

[3]. R. T. Born and R. B. H. Tootell, "Segregation of global and local motion processing in primate middle temporal visual area," *Nature*, vol. 357, pp. 497-499, 1992.

[4]. J. Allman, F. Miezin and E. McGuinness, "Stimulus specific responses from beyond the classical receptive field: Neurophysiological mechanisms for local-global comparisions in visual neurons," *Annual Review Neuroscience*, vol. 8, pp. 407-430, 1985.

[5]. V. A. F. Lamme, "The neurophysiology of figure-ground segregation in primary visual cortex," *Journal of Neuroscience*, vol. 15, pp. 1605-1615, 1995.

[6]. D. C. Bradley and R. A. Andersen, "Center-surround antagonism based on disparity in primate area MT," *Journal of Neuroscience*, vol. 18, pp. 7552-65, 1998.

[7]. D. J. Heeger, A. D. Jepson and E. P. Simoncelli, "Recovering observer translation with center-surround operators," *Proc IEEE Workshop on Visual Motion*, pp. 95-100, Oct 1991.

[8]. N. Petkov and E. Subramanian, "Motion detection, noise reduction, texture suppression, and contour enhancement by spatiotemporal Gabor filters with surround inhibition," *Biological Cybernetics*, vol. 97, pp. 423-439, 2007.

[9]. M. Escobar and P. Kornprobst, "Action recognition with a Bio-inspired feedforward motion processing model: the richness of center-surround interactions," ECCV '08: *Proceedings of the 10th European Conference on Computer Vision*, pp. 186-199, Marseille, France, 2008.

[10]. B. J. Frost and K. Nakayama, "Single visual neurons code opposing motion independent of direction," *Science*, vol. 200, pp. 744-745, 1983.

[11]. A. Cao and P. H. Schiller, "Neural responses to relative speed in the primary visual cortex of rhesus monkey," *Visual Neuroscience*, vol. 20, pp. 77-84, 2003.

[12]. B. K. Dellen, J. W. Clark and R. Wessel, "Computing relative motion with complex cells," *Visual Neuroscience*, vol. 22, pp. 225-236, 2005.

[13]. I. Ohzawa, G. C. Deangelis and R. D. Freeman, "Encoding of binocular disparity by complex cells in the cat's visual cortex," *Journal of Neurophysiology*, vol. 77, pp. 2879-2909, 1997.

[14]. D. J. Fleet, H. Wagner and D. J. Heeger, "Neural Encoding of binocular disparity: energy model, position shifts and phase shifts," *Vision Research*, vol. 26, pp. 1839-1857, 1996.

[15]. Y. C. Meng and B. E. Shi, "Normalized Phase Shift Motion Energy Neuron Populations for Image Velocity Estimation," *International Joint Conference on Neural Network*, Atlanta, GA, June 14-19, 2009.

[16]. E. H. Adelson and J. R. Bergen, "Spatiotemporal energy models for the perception of motion," *J. Opt. Soc. Am. A Opt. Image Sci. Vis.*, vol. 2, pp. 284-299, 1985.

[17]. D. K. Xiao, S. Raiguel, V. Marcar, J. Koenderink and G. A. Orban, "The spatial distribution of the antagonistic surround of MT/V5," *Cereb Cortex*, vol. 7, pp. 662-677, 1997.

[18]. D. Tadin, J. S. Lappin, L. A. Gilroy and R. Blake, "Perceptual consequences of centre-surround antagonism in visual motion processing," *Nature*, vol. 424, pp. 312-315, 2003.

[19]. X. Huang, T. D. Albright and G. R. Stoner, "Adaptive surround modulation in cortical area MT," *Neuron*, vol. 53, pp. 761-770, 2007.
